# Bayesian model selection for Support Vector machines, Gaussian processes and other kernel classifiers

**Matthias Seeger**
Institute for Adaptive and Neural Computation
University of Edinburgh
5 Forrest Hill, Edinburgh EH1 2QL
*seeger@dai.ed.ac.uk*

## Abstract

We present a variational Bayesian method for model selection over families of kernels classifiers like Support Vector machines or Gaussian processes. The algorithm needs no user interaction and is able to adapt a large number of kernel parameters to given data without having to sacrifice training cases for validation. This opens the possibility to use sophisticated families of kernels in situations where the small "standard kernel" classes are clearly inappropriate. We relate the method to other work done on Gaussian processes and clarify the relation between Support Vector machines and certain Gaussian process models.

## 1 Introduction

Bayesian techniques have been widely and successfully used in the neural networks and statistics community and are appealing because of their conceptual simplicity, generality and consistency with which they solve learning problems. In this paper we present a new method for applying the Bayesian methodology to Support Vector machines. We will briefly review Gaussian Process and Support Vector classification in this section and clarify their relationship by pointing out the common roots. Although we focus on classification here, it is straightforward to apply the methods to regression problems as well. In section 2 we introduce our algorithm and show relations to existing methods. Finally, we present experimental results in section 3 and close with a discussion in section 4.

Let $X$ be a measure space (e.g. $X = \mathbb{R}^d$) and $D = (\mathcal{X}, t) = \{(x_1, t_1), \ldots, (x_n, t_n)\}$, $x_i \in X$, $t_i \in \{-1, +1\}$ a noisy i.i.d. sample from a *latent function* $y : X \to \mathbb{R}$, where $P(t|y)$ denotes the noise distribution. Given further points $x_*$ we wish to predict $t_*$ so as to minimize the error probability $P(t|x_*, D)$, or (more difficult) to estimate this probability. *Generative Bayesian* methods attack this problem by placing a stochastic process prior $P(y(\cdot))$ over the space of latent functions and

then compute *posterior* and *predictive* distributions $P(y|D)$, $P(y_*|x_*, D)$ as

$$P(y|D) = \frac{P(D|y)P(y)}{P(D)},$$

$$P(y_*|D, x_*) = \int P(y_*|y)P(y|D)\, dy \tag{1}$$

where $y = (y(x_i))_i$, $y_* = y(x_*)$, the likelihood $P(D|y) = \prod_i P(t_i|y_i)$ and $P(D)$ is a normalization constant. $P(t|x_*, D)$ can then be obtained by averaging $P(t|y_*)$ over $P(y_*|x_*, D)$. *Gaussian process (GP)* or *spline smoothing* models use a Gaussian process prior on $y(\cdot)$ which can be seen as function of $X$ into a set of random variables such that for each finite $X_1 \subset X$ the corresponding variables are jointly Gaussian (see [15] for an introduction). A GP is determined by a mean function[1] $x \mapsto \mathrm{E}\,[y(x)]$ and a positive definite *covariance kernel* $K(x, \tilde{x})$. *Gaussian process classification (GPC)* amounts to specifying available prior knowledge by choosing a class of kernels $K(x, x|\theta)$, $\theta \in \Theta$, where $\theta$ is a vector of *hyperparameters*, and a *hyperprior* $P(\theta)$. Usually, these choices are guided by simple attributes of $y(\cdot)$ such as smoothness, trends, differentiability, but more general approaches to kernel design have also been considered [5]. For 2-class classification the most common noise distribution is the *binomial* one where $P(t|y) = \sigma(ty)$, $\sigma(u) = (1 + \exp(-u))^{-1}$ the *logistic* function, and $y$ is the *logit* $\log(P(+1|x)/P(-1|x))$ of the target distribution. For this noise model the integral in (1) is not analytically tractable, but a range of approximative techniques based on *Laplace approximations* [16], *Markov chain Monte Carlo* [7], variational methods [2] or mean field algorithms [8] are known.

We follow [16]. The Laplace approach to GPC is to approximate the posterior $P(y|D, \theta)$ by the Gaussian distribution $N(\hat{y}, \mathcal{H}^{-1})$ where $\hat{y} = \operatorname{argmax} P(y|D, \theta)$ is the posterior mode and $\mathcal{H} = \nabla'_y \nabla_y (-\log P(y|D, \theta))$, evaluated at $\hat{y}$. Then it is easy to show that the predictive distribution is Gaussian with mean $k(x_*)'\mathcal{K}^{-1}\hat{y}$ and variance $k_* - k(x_*)'\mathcal{K}^{-1}k(x_*)$ where $\mathcal{K}$ is the covariance matrix $(K(x_i, x_j))_{ij}$, $k(\cdot) = (K(x_i, \cdot))_i$, $k_* = K(x_*, x_*)$ and the prime denotes transposition. The final discriminant is therefore a linear combination of the $K(x_i, \cdot)$.

The *discriminative* approach to the prediction problem is to choose a *loss function* $g(t, y)$, being an approximation to the *misclassification loss*[2] $I_{\{ty \le 0\}}$ and then to search for a discriminant $y(\cdot)$ which minimizes $\mathrm{E}\,[g(t, y(x_*))]$ for the points $x_*$ of interest (see [14]). *Support Vector classification (SVC)* uses the $\varepsilon$-insensitive loss (SVC loss) $g(t, y) = [1 - ty]_+$, $[u]_+ = uI_{\{u \ge 0\}}$ which is an upper bound on the misclassification loss, and a *reproducing kernel Hilbert space (RKHS)* with kernel $K(x, \tilde{x}|\theta)$ as hypothesis space for $y(\cdot)$. Indeed, Support Vector models and the Laplace method for Gaussian processes are special cases of *spline smoothing* models in RKHS where the aim is to minimize the functional

$$\sum_{i=1}^{n} g(t_i, y_i) + \lambda\|y(\cdot)\|_K^2 \tag{2}$$

where $\|\cdot\|_K$ denotes the norm of the RKHS. It can be shown that the minimizer of (2) can be written as $k(\cdot)'\mathcal{K}^{-1}\hat{y}$ where $\hat{y}$ maximizes

$$-\sum_{i=1}^{n} g(t_i, y_i) - \lambda y'\mathcal{K}^{-1}y. \tag{3}$$

All these facts can be found in [13]. Now (3) is, up to terms not depending on $y$, the log posterior in the above GP framework if we choose $g(t, y) = -\log P(t|y)$ and

absorb $\lambda$ into $\boldsymbol{\theta}$. For the SVC loss, (3) can be transformed into a dual problem via $\boldsymbol{y} = \mathcal{K}\boldsymbol{\alpha}$, where $\boldsymbol{\alpha}$ is a vector of dual variables, which can be efficiently solved using quadratic programming techniques. [12] is an excellent reference.

Note that the SVC loss cannot be written as the negative log of a noise distribution, so we cannot reduce SVC to a special case of a Gaussian process classification model. Although a generative model for SVC is given in [11], it is easier and less problematic to regard SVC as efficient approximation to a proper Gaussian process model. Various such models have been proposed (see [8],[4]). In this work, we simply normalize the SVC loss pointwise, i.e. use a Gaussian process model with the *normalized SVC loss* $g(t, y) = [1 - ty]_+ + \log Z(y)$, $Z(y) = \exp(-[1 - y]_+) + \exp(-[1 + y]_+)$. Note that $g(t, y)$ is a close approximation of the (unnormalized) SVC loss. The reader might miss the SVM *bias parameter* which we dropped here for clarity, but it is straightforward to apply this *semiparametric* extension to GP models too[3].

## 2  A variational method for kernel classification

The real Bayesian way to deal with the hyperparameters $\boldsymbol{\theta}$ is to average $P(y_*|\boldsymbol{x}_*, D, \boldsymbol{\theta})$ over the posterior $P(\boldsymbol{\theta}|D)$ in order to obtain the predictive distribution $P(y_*|\boldsymbol{x}_*, D)$. This can be approximated by Markov chain Monte Carlo methods [7], [16] or simply by $P(y_*|\boldsymbol{x}_*, D, \hat{\boldsymbol{\theta}})$, $\hat{\boldsymbol{\theta}} = \arg\max P(\boldsymbol{\theta}|D)$. The latter approach, called *maximum a-posteriori (MAP)*, can be justified in the limit of large $n$ and often works well in practice. The basic challenge of MAP is to calculate the *evidence*

$$P(D|\boldsymbol{\theta}) = \int P(D, \boldsymbol{y}|\boldsymbol{\theta})\, d\boldsymbol{y} = \int \exp\left(-\sum_{i=1}^{n} g(t_i, y_i)\right) N(\boldsymbol{y}|0, \mathcal{K}(\boldsymbol{\theta}))\, d\boldsymbol{y}. \quad (4)$$

Our plan is to attack (4) by a *variational* approach. Let $\tilde{P}$ be a density from a model class $\Gamma$ chosen to approximate the posterior $P(\boldsymbol{y}|D, \boldsymbol{\theta})$. Then:

$$
\begin{aligned}
-\log P(D|\boldsymbol{\theta}) &= -\int \tilde{P}(\boldsymbol{y}) \log\left(\frac{P(D, \boldsymbol{y}|\boldsymbol{\theta})\tilde{P}(\boldsymbol{y})}{P(\boldsymbol{y}|D, \boldsymbol{\theta})\tilde{P}(\boldsymbol{y})}\right) d\boldsymbol{y} \\
&= F(\tilde{P}, \boldsymbol{\theta}) - \int \tilde{P}(\boldsymbol{y}) \log\left(\frac{\tilde{P}(\boldsymbol{y})}{P(\boldsymbol{y}|D, \boldsymbol{\theta})}\right) d\boldsymbol{y}
\end{aligned}
\quad (5)
$$

where we call $F(\tilde{P}, \boldsymbol{\theta}) = \mathrm{E}_{\tilde{P}}[-\log P(D, \boldsymbol{y}|\boldsymbol{\theta})] + \mathrm{E}_{\tilde{P}}[\log \tilde{P}(\boldsymbol{y})]$ the *variational free energy*. The second term in (5) is the well-known *Kullback-Leibler divergence* between $\tilde{P}$ and the posterior which is nonnegative and equals zero iff $\tilde{P}(\boldsymbol{y}) = P(\boldsymbol{y}|D, \boldsymbol{\theta})$ almost everywhere with respect to the distribution $\tilde{P}$. Thus, $F$ is an upper bound on $-\log P(D|\boldsymbol{\theta})$, and changing $(\tilde{P}, \boldsymbol{\theta})$ to decrease $F$ enlarges the evidence or decreases the divergence between the posterior and its approximation, both being favourable. This idea has been introduced in [3] as *ensemble learning*[4] and has been successfully applied to MLPs [1]. The latter work also introduced the model class $\Gamma$ we use here, namely the class of Gaussians with mean $\boldsymbol{\mu}$ and *factor-analyzed* covariance $\Sigma = \mathcal{D} + \sum_{j=1}^{M} \boldsymbol{c}_j \boldsymbol{c}_j'$, $\mathcal{D}$ diagonal with positive elements[5]. Hinton and

van Camp [3] used diagonal covariances which would be $M = 0$ in our setting. By choosing a small $M$, we are able to track the most important correlations between the components in the posterior using $O(Mn)$ parameters to represent $\tilde{P}$.

Having agreed on $\Gamma$, the criterion $F$ and its gradients with respect to $\theta$ and the parameters of $\tilde{P}$ can easily and efficiently be computed except for the generic term

$$\mathrm{E}_{\tilde{P}} \left[ \sum_{i=1}^{n} g(t_i, y_i) \right], \tag{6}$$

a sum of one-dimensional Gaussian expectations which are, depending on the actual $g$, either analytically tractable or can be approximated using a quadrature algorithm. For example, the expectation for the normalized SVC loss can be decomposed into expectations over the (unnormalized) SVC loss and over $\log Z(y)$ (see end of section 1). While the former can be computed analytically, the latter expectation can be handled by replacing $\log Z(y)$ by a piecewise defined tight bound such that the integral can be solved analytically. For the GPC loss (6) cannot be solved analytically and was in our experiments approximated by Gaussian quadrature.

We can optimize $F$ using a nested loop algorithm as follows. In the inner loop we run an optimizer to minimize $F$ w.r.t. $\tilde{P}$ for fixed $\theta$. We used a *conjugate gradients* optimizer since the number of parameters of $\tilde{P}$ is rather large. The outer loop is an optimizer minimizing $F$ w.r.t. $\theta$, and we chose a *Quasi-Newton* method here since the dimension of $\Theta$ is usually rather small and gradients w.r.t. $\theta$ are costly to evaluate.

We can use the resulting minimizer $(\tilde{P}, \hat{\theta})$ of $F$ in two different ways. The most natural is to discard $\tilde{P}$, plug $\hat{\theta}$ into the original architecture and predict using the mode of $P(y|D, \hat{\theta})$ as an approximation to the true posterior mode, benefitting from a kernel now adapted to the given data. This is particularly interesting for Support Vector machines due to the sparseness of the final kernel expansion (typically only a small fraction of the components in the weight vector $\mathcal{K}^{-1}\hat{y}$ is non-zero, the corresponding datapoints are termed *Support Vectors*) which allows very efficient predictions for a large number of test points. However, we can also retain $\tilde{P}$ and use it as a Gaussian approximation of the posterior $P(y|D, \hat{\theta})$. Doing so, we can use the variance of the approximative predictive distribution $P(y_*|x_*, D)$ to derive *error bars* for our predictions, although the interpretation of these figures is somewhat complicated in the case of kernel discriminants like SVM whose loss function does not correspond to a noise distribution.

## 2.1   Relations to other methods

Let us have a look at alternative ways to maximize (4). If the loss $g(t, y)$ is twice differentiable everywhere, progress can be made by replacing $g$ by its second order Taylor expansion around the mode of the integrand. This is known as *Laplace approximation* and is used in [16] to maximize (4) approximately. However, this technique cannot be used for nondifferentiable losses of the $\varepsilon$-insensitive type[6].

Nevertheless, for the SVC loss the evidence (4) can be approximated in a Laplace-like fashion [11], and it will be interesting to compare the results of this work with ours. This approximation can be evaluated very efficiently, but is not continuous[7]

w.r.t. $\boldsymbol{\theta}$ and difficult to optimize if the dimension of $\Theta$ is not small. Opper and Winther [8] use mean field ideas to derive an approximate leave-one-out test error estimator which can be quickly evaluated, but suffers from the typical noisiness of cross-validation scores. Kwok [6] applies the evidence framework to Support Vector machines, but the technique seems to be restricted to kernels with a finite eigenfunction expansion (see [13] for details).

It is interesting to compare our variational method to the Laplace method of [16] and the variational technique of [2]. Let $g(t, y)$ be differentiable and suppose that for given $\boldsymbol{\theta}$ we restrict ourselves to approximate (6) by replacing $g(t_i, y_i)$ by the expansion

$$g(t_i, \mu_i) + \frac{\partial g}{\partial y}(t_i, \mu_i)(y_i - \mu_i) + \frac{1}{2}\frac{\partial^2 g}{\partial y^2}(t_i, \hat{y}_i)(y_i - \mu_i)^2, \tag{7}$$

where $\hat{y}$ is the posterior mean. This will change the criterion $F$ to $F_{approx}$, say. Then it is easy to show that the Gaussian approximation to the posterior employed by the Laplace method, namely $N(\hat{y}, (\mathcal{K}^{-1} + \mathcal{W})^{-1})$, $\mathcal{W} = \mathrm{diag}(\sigma(\hat{y}_i)(1 - \sigma(\hat{y}_i)))$, minimizes $F_{approx}$ w.r.t. $\tilde{P}$ if full covariances $\Sigma$ are used, and plugging this minimizer into $F_{approx}$ we end up with the evidence approximation which is maximized by the Laplace method. The latter is not a variational technique since the approximation (7) to the loss function is not an upper bound, and works only for differentiable loss functions. If we upper bound the loss function $g(t, y)$ by a quadratic polynomial and add the variational parameters of this bound to the parameters of $\tilde{P}$, our method becomes broadly similar to the lower bound algorithm of [2]. Indeed, since for fixed variational parameters of the polynomials we can easily solve for the mean and covariance of $\tilde{P}$, the former parameters are the only essential ones. However, the quadratic upper bound is poor for functions like the SVC loss, and in these cases our bound is expected to be tighter.

## 3 Experiments

We tested our variational algorithm on a number of datasets from the *UCI machine learning repository* and the *DELVE archive* of the University of Toronto[8]: *Leptograpsus crabs*, *Pima Indian diabetes*, *Wisconsin Breast Cancer*, *Ringnorm*, *Twonorm* and *Waveform* (class 1 against 2). Descriptions may be found on the web. In each case we normalized the whole set to zero mean, unit variance in all input columns, picked a training set at random and used the rest for testing. We chose (for $X = \mathbb{R}^d$) the well-known *squared-exponential kernel* (see [15]):

$$K(\boldsymbol{x}, \tilde{\boldsymbol{x}}|\boldsymbol{\theta}) = C\left(\exp\left(-\frac{1}{2d}\sum_{i=1}^{d} w_i(x_i - \tilde{x}_i)^2\right) + v\right), \quad \boldsymbol{\theta} = ((w_i)_i', C, v)'. \tag{8}$$

All parameters are constrained to be positive, so we chose the representation $\theta_i = \nu_i^2$. We did not use a prior on $\boldsymbol{\theta}$ (see comment at end of this section). For comparison we trained a Gaussian Process classifier with the Laplace method (also without hyperprior) and a Support Vector machine using 10-fold cross-validation to select the free parameters. In the latter case we constrained the scale parameters $w_i$ to be equal (it is infeasible to adapt $d + 2$ hyperparameters to the data using cross-validation) and dropped the $v$ parameter while allowing for a bias parameter. As mentioned above, within the variational method we can use the posterior mode $\hat{y}$

http://www.ics.uci.edu/~mlearn/MLRepository.html.

| Name | train size | test size | Var. GP $\hat{y}$ | $\mu$ | GP Lapl. | Var. SVM $\hat{y}$ | $\mu$ | SVM 10-CV | Lin. discr. |
|------|-----------|-----------|-----------|-------|----------|-----------|-------|-----------|-------------|
| crabs | 80 | 120 | 3 | 4 | 4 | 4 | 4 | 4 | 3 |
| pima | 200 | 332 | 66 | 66 | 68 | 64 | 66 | 67 | 67 |
| wdbc | 300 | 269 | 11 | 11 | 8 | 10 | 10 | 9 | 19 |
| twonorm | 300 | 7100 | 233 | 224 | 297 | 260 | 223 | 163 | 207 |
| ringnorm | 400 | 7000 | 119 | 124 | 184 | 129 | 126 | 160 | 1763 |
| waveform | 800 | 2504 | 206 | 204 | 221 | 211 | 206 | 197 | 220 |

Table 1: Number of test errors for various methods.

as well as the mean $\mu$ of $\tilde{P}$ for prediction, and we tested both methods. Error bars were not computed. The baseline method was a linear discriminant trained to minimize the squared error. Table 1 shows the test errors the different methods attained.

These results show that the new algorithm performs equally well as the other methods we considered. They have of course to be regarded in combination with how much effort was necessary to produce them. It took us almost a whole day and a lot of user interactions to do the cross-validation model selection. The rule-of-thumb that a lot of Support Vectors at the upper bound indicate too large a parameter $C$ in (8) failed for at least two of these sets, so we had to start with very coarse grids and sweep through several stages of refinement.

An effect known as *automatic relevance determination (ARD)* (see [7]) can be nicely observed on some of the datasets, by monitoring the length scale parameters $w_i$ in (8). Indeed, our variational SVC algorithm almost completely ignored (by driving their length scales to very small values) 3 of the 5 dimensions in "crabs", 2 of 7 in "pima" and 3 of 21 in "waveform". On "wdbc", it detected dimension 24 as particularly important with regard to separation, all this in harmony with the GP Laplace method. Thus, a sensible parameterized kernel family together with a method of the Bayesian kind allows us to gain additional important information from a dataset which might be used to improve the experimental design.

Results of experiments with the methods tested above and hyperpriors as well as a more detailed analysis of the experiments can be found in [9].

## 4   Discussion

We have shown how to perform model selection for Support Vector machines using approximative Bayesian variational techniques. Our method is applicable to a wide range of loss functions and is able to adapt a large number of hyperparameters to given data. This allows for the use of sophisticated kernels and Bayesian techniques like *automatic relevance determination* (see [7]) which is not possible using other common model selection criteria like *cross-validation*. Since our method is fully automatic, it is easy for non-experts to use[9], and as the evidence is computed on the training set, no training data has to be sacrificed for validation. We refer to [9] where the topics of this paper are investigated in much greater detail.

A pressing issue is the unfortunate scaling of the method with the training set

size $n$ which is currently $O(n^3)$[10]. We are currently exploring the applicability of the powerful approximations of [10] which might bring us very much closer to the desired $O(n^2)$ scaling (see also [2]). Another interesting issue would be to connect our method with the work of [5] who use generative models to derive kernels in situations where the "standard kernels" are not applicable or not reasonable.

## Acknowledgments

We thank Chris Williams, Amos Storkey, Peter Sollich and Carl Rasmussen for helpful and inspiring discussions. This work was partially funded by a scholarship of the *Dr. Erich Müller foundation*. We are grateful to the Division of Informatics for supporting our visit in Edinburgh, and to Chris Williams for making it possible.

## Footnotes

[1]W.l.o.g. we only consider GPs with mean function 0 in what follows.

[2]$I_A$ denotes the indicator function of the set $A \subset \mathbb{R}$.

[3] This is the "random effects model with improper prior" of [13], p.19, and works by placing a flat improper prior on the bias parameter.

[4] We average different discriminants (given by $\boldsymbol{y}$) over the ensemble $\tilde{P}$.

[5] Although there is no danger of overfitting, the use of full covariances would render the optimization more difficult, time and memory consuming.

[6]The nondifferentiabilities cannot be ignored since with probability one a nonzero number of the $\hat{y}_i$ sit exactly at these margin locations.

[7]Although continuity can be accomplished by a further modification, see [11].

[8]See http://www.cs.utoronto.ca/~delve and

[9]As an aside, this opens the possibility of comparing SVMs against other fully-automatic methods within the *DELVE* project (see section 3).

[10]The running time is essentially the same as that of the Laplace method, thus being comparable to the fastest known Bayesian GP algorithm.

# References

[1] David Barber and Christopher Bishop. Ensemble learning for multi-layer networks. In *Advances in NIPS*, number 10, pages 395–401. MIT Press, 1997.

[2] Mark N. Gibbs. *Bayesian Gaussian Processes for Regression and Classification.* PhD thesis, University of Cambridge, 1997.

[3] Geoffrey E. Hinton and D. Van Camp. Keeping neural networks simple by minimizing the description length of the weights. In *Proceedings of the 6th annual conference on computational learning theory*, pages 5–13, 1993.

[4] Tommi Jaakkola, Marina Meila, and Tony Jebara. Maximum entropy discrimination. In *Advances in NIPS*, number 13. MIT Press, 1999.

[5] Tommi S. Jaakkola and David Haussler. Exploiting generative models in discriminative classifiers. In *Advances in NIPS*, number 11, 1998.

[6] James Tin-Tau Kwok. Integrating the evidence framework and the Support Vector machine. Submitted to ESANN 99, 1999.

[7] Radford M. Neal. Monte Carlo implementation of Gaussian process models for Bayesian classification and regression. Technical Report 9702, Department of Statistics, University of Toronto, January 1997.

[8] Manfred Opper and Ole Winther. GP classification and SVM: Mean field results and leave-one-out estimator. In *Advances in Large Margin Classifiers*. MIT Press, 1999.

[9] Matthias Seeger. Bayesian methods for Support Vector machines and Gaussian processes. Master's thesis, University of Karlsruhe, Germany, 1999. Available at http://www.dai.ed.ac.uk/~seeger.

[10] John Skilling. *Maximum entropy and Bayesian methods*. Cambridge University Press, 1988.

[11] Peter Sollich. Probabilistic methods for Support Vector machines. In *Advances in NIPS*, number 13. MIT Press, 1999.

[12] Vladimir N. Vapnik. *Statistical Learning Theory*. Wiley, 1998.

[13] Grace Wahba. *Spline Models for Observational Data*. CBMS-NSF Regional Conference Series. SIAM, 1990.

[14] Grace Wahba. Support Vector machines, reproducing kernel Hilbert spaces and the randomized GACV. Technical Report 984, University of Wisconsin, 1997.

[15] Christopher K. I. Williams. Prediction with Gaussian processes: From linear regression to linear prediction and beyond. In M. I. Jordan, editor, *Learning in Graphical Models*. Kluwer, 1997.

[16] Christopher K.I. Williams and David Barber. Bayesian classification with Gaussian processes. *IEEE Trans. PAMI*, 20(12):1342–1351, 1998.

